# Tree-Based Modeling and Estimation of Gaussian Processes on Graphs with Cycles

**Martin J. Wainwright, Erik B. Sudderth, and Alan S. Willsky**
Laboratory for Information and Decision Systems
Department of Electrical Engineering and Computer Science
Massachusetts Institute of Technology
Cambridge, MA 02139
{*mjwain,esuddert,willsky*}*@mit.edu*

## Abstract

We present the embedded trees algorithm, an iterative technique for estimation of Gaussian processes defined on arbitrary graphs. By exactly solving a series of modified problems on embedded spanning trees, it computes the conditional means with an efficiency comparable to or better than other techniques. Unlike other methods, the embedded trees algorithm also computes exact error covariances. The error covariance computation is most efficient for graphs in which removing a small number of edges reveals an embedded tree. In this context, we demonstrate that sparse loopy graphs can provide a significant increase in modeling power relative to trees, with only a minor increase in estimation complexity.

## 1 Introduction

Graphical models are an invaluable tool for defining and manipulating probability distributions. In modeling stochastic processes with graphical models, two basic problems arise: (i) specifying a class of graphs with which to model or approximate the process; and (ii) determining efficient techniques for statistical inference. In fact, there exists a fundamental tradeoff between the expressive power of a graph, and the tractability of statistical inference. At one extreme are tree-structured graphs: although they lead to highly efficient algorithms for estimation [1, 2], their modeling power is often limited. The addition of edges to the graph tends to increase modeling power, but also introduces loops that necessitate the use of more sophisticated and costly techniques for estimation.

In areas like coding theory, artificial intelligence, and speech processing [3, 1], graphical models typically involve discrete-valued random variables. However, in domains such as image processing, control, and oceanography [2, 4, 5], it is often more appropriate to consider random variables with a continuous distribution. In this context, Gaussian processes on graphs are of great practical significance. Moreover, the Gaussian case provides a valuable setting for developing an understanding of estimation algorithms [6, 7].

The focus of this paper is the estimation and modeling of Gaussian processes defined on graphs with cycles. We first develop an estimation algorithm that is based on exploiting trees embedded within the loopy graph. Given a set of noisy measurements, this embedded trees (ET) algorithm computes the conditional means with an efficiency comparable to or better than other techniques. Unlike other methods, the ET algorithm also computes exact error covariances at each node. In many applications, these error statistics are as important as the conditional means. We then demonstrate by example that relative to tree models, graphs with a small number of loops can lead to substantial improvements in modeling fidelity without a significant increase in estimation complexity.

## 2 Linear estimation fundamentals

### 2.1 Problem formulation

Consider a Gaussian stochastic process $x \sim \mathcal{N}(0, P)$ that is Markov with respect to an undirected graph $\mathcal{G}$. Each node in $\mathcal{G}$ corresponds to a subvector $x_i$ of $x$. We will refer to $x_i$ as the state variable for the $i^{th}$ node, and its length as the state dimension. By the Hammersley–Clifford Theorem [8], $P^{-1}$ inherits a sparse structure from $\mathcal{G}$. If it is partitioned into blocks according to the state dimensions, the $(i, j)^{th}$ block can be nonzero only if there is an edge between nodes $i$ and $j$.

Let $y = Cx + v$, $v \sim \mathcal{N}(0, R)$, be a set of noisy observations. Without loss of generality, we assume that the subvectors $y_i$ of the observations are conditionally independent given the state $x$. For estimation purposes, we are interested in $p(x_i|y)$, the marginal distribution of the state at each node conditioned on the noisy observations. Standard formulas exist for the computation of $p(x|y) \sim \mathcal{N}(\widehat{x}, \widehat{P})$:

$$\widehat{x} = \widehat{P} \, C^T R^{-1} y \qquad \widehat{P} = [P^{-1} + C^T R^{-1} C]^{-1} \qquad (1)$$

The conditional error covariances $\widehat{P}_i$ are the block diagonal elements of the full error covariance $\widehat{P}$, where the block sizes are equal to the state dimensions.

### 2.2 Exploiting graph structure

When $\mathcal{G}$ is tree structured, both the conditional means and error covariances can be computed by a direct and very efficient $\mathcal{O}(d^3 N)$ algorithm [2]. Here $d$ is the maximal state dimension at any node, and $N$ is the total number of nodes. This algorithm is a generalization of classic Kalman smoothing algorithms for time series, and involves passing means and covariances to and from a node chosen as the root.

For graphs with cycles, calculating the full error covariance $\widehat{P}$ by brute force matrix inversion would, in principle, provide the conditional means and error variances. Since the computational complexity of matrix inversion is $\mathcal{O}([dN]^3)$, this proposal is not practically feasible in many applications, such as image processing, where $N$ may be on the order of $10^5$. This motivates the development of iterative techniques for linear estimation on graphs with cycles.

Recently, two groups [6, 7] have analyzed Pearl's belief propagation [1] in application to Gaussian processes defined on loopy graphs. For Gaussians on trees, belief propagation produces results equivalent to the Kalman smoother of Chou et al. [2]. For graphs with cycles, these groups showed that when belief propagation converges, it computes the correct conditional means, but that error covariances are incorrect. The complexity per iteration of belief propagation on loopy graphs is $\mathcal{O}(d^3 N)$, where one iteration corresponds to updating each message once.

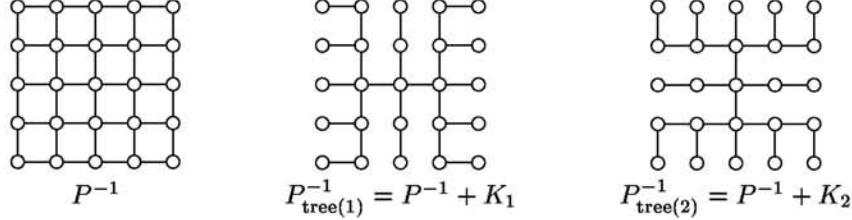

$$P^{-1} \qquad P_{\text{tree}(1)}^{-1} = P^{-1} + K_1 \qquad P_{\text{tree}(2)}^{-1} = P^{-1} + K_2$$

**Figure 1.** Embedded trees produced by two different cutting matrices $K_i$ for a nearest–neighbor grid (observation nodes not shown).

It is important to note that conditional means can be efficiently calculated using techniques from numerical linear algebra [9]. In particular, it can be seen from equation (1) that computing the conditional mean $\hat{x}$ is equivalent to computing the product of a matrix inverse and a vector. Given the sparsity of $P^{-1}$, iterative techniques like conjugate gradient [9] can be used to compute the mean with associated cost $\mathcal{O}(dN)$ per iteration. However, like belief propagation, such techniques compute only the means and not the error covariances.

## 3   Embedded trees algorithm

### 3.1   Calculation of means

In this section, we present an iterative algorithm for computing both the conditional means and error covariances of a Gaussian process defined on any graph. Central to the algorithm is the operation of cutting edges from a loopy graph to reveal an embedded tree. Standard tree algorithms [2] can be used to exactly solve the modified problem, and the results are used in a subsequent iteration.

For a Gaussian process on a graph, the operation of removing edges corresponds to modifying the inverse covariance matrix. Specifically, we apply a matrix splitting

$$P^{-1} + C^T R^{-1} C = P_{\text{tree}(t)}^{-1} - K_t + C^T R^{-1} C$$

where $K_t$ is a symmetric cutting matrix chosen to ensure that $P_{\text{tree}(t)}^{-1}$ corresponds to a valid tree-structured inverse covariance matrix. This matrix splitting allows us to consider defining a sequence of iterates $\{\hat{x}^n\}$ by the recursion:

$$\left[ P_{\text{tree}(t(n))}^{-1} + C^T R^{-1} C \right] \hat{x}^n = K_{t(n)} \hat{x}^{n-1} + C^T R^{-1} y$$

Here $t(n)$ indexes the embedded tree used in the $n^{th}$ iteration. For example, Figure 1 shows two of the many spanning trees embedded in a nearest–neighbor grid. When the matrix $(P_{\text{tree}(t(n))}^{-1} + C^T R^{-1} C)$ is positive definite, it is possible to solve for the next iterate $\hat{x}^n$ in terms of data $y$ and the previous iterate. Thus, given some starting point $\hat{x}^0$, we can generate a sequence of iterates $\{\hat{x}^n\}$ by the recursion

$$\hat{x}^n = M_{t(n)}^{-1} \left[ K_{t(n)} \hat{x}^{n-1} + C^T R^{-1} y \right] \qquad (2)$$

where $M_{t(n)} \triangleq (P_{\text{tree}(t(n))}^{-1} + C^T R^{-1} C)$. By comparing equation (2) to equation (1), it can be seen that computing the $n^{th}$ iterate corresponds to a linear-Gaussian problem, which can be solved efficiently and directly with standard tree algorithms [2].

### 3.2   Convergence of means

Before stating some convergence results, recall that for any matrix $A$, the spectral radius is defined as $\rho(A) \triangleq \max_\lambda |\lambda|$, where $\lambda$ ranges over the eigenvalues of $A$.

**Proposition 1.** Let $\widehat{x}$ be the conditional mean of the original problem on the loopy graph, and consider the sequence of iterates $\{\widehat{x}^n\}$ generated by equation (2). Then for any starting point, $\widehat{x}$ is the unique fixed point of the recursion, and the error $e^n \triangleq \widehat{x}^n - \widehat{x}$ obeys the dynamics

$$e^n = \left[ \prod_{j=1}^{n} M_{\text{tree}(t(j))}^{-1} K_{t(j)} \right] e^0 \tag{3}$$

In a typical implementation of the algorithm, one cycles through the embedded trees in some fixed order, say $t = 1, \ldots, T$. In this case, the convergence of the algorithm can be analyzed in terms of the product matrix $\mathbf{A} \triangleq \prod_{j=1}^{T} M_{\text{tree}(j)}^{-1} K_j$.

**Proposition 2.** Convergence of the ET algorithm is governed by the spectral radius of $\mathbf{A}$. In particular, if $\rho(\mathbf{A}) > 1$, then the algorithm will not converge, whereas if $\rho(\mathbf{A}) < 1$, then $(\widehat{x}^n - \widehat{x}) \stackrel{n \to \infty}{\longrightarrow} 0$ geometrically at rate $\gamma \triangleq \rho(\mathbf{A})^{\frac{1}{T}}$.

Note that the cutting matrices $K$ must be chosen in order to guarantee not only that $P_{\text{tree}}^{-1}$ is tree-structured but also that $M \triangleq (P_{\text{tree}}^{-1} + C^T R^{-1} C)$ is positive definite. The following theorem, adapted from results in [10], gives conditions guaranteeing the validity and convergence of the ET algorithm when cutting to a single tree.

**Theorem 1.** Define $Q \triangleq P^{-1} + C^T R^{-1} C$, and $M \triangleq Q + K$. Suppose the cutting matrix $K$ is symmetric and positive semidefinite. Then we are guaranteed that $\rho(M^{-1} K) < 1$. In particular, we have the bounds:

$$\frac{\lambda_{max}(K)}{\lambda_{max}(K) + \lambda_{max}(Q)} \leq \rho(M^{-1} K) \leq \frac{\lambda_{max}(K)}{\lambda_{max}(K) + \lambda_{min}(Q)} \tag{4}$$

It should be noted that the conditions of this theorem are sufficient, but by no means necessary, to guarantee convergence of the ET algorithm. In particular, we find that indefinite cutting matrices often lead to faster convergence. Furthermore, Theorem 1 does not address the superior performance typically achieved by cycling through several embedded trees. Gaining a deeper theoretical understanding of these phenomena is an interesting open question.

### 3.3 Calculation of error covariances

Although there exist a variety of iterative algorithms for computing the conditional mean of a linear-Gaussian problem, none of these methods correctly compute error covariances at each node. We show here that the ET algorithm can efficiently compute these covariances in an iterative fashion. For many applications (e.g., oceanography [5]), these error statistics are as important as the estimates.

We assume for simplicity in notation that $\widehat{x}^0 = 0$ and then expand equation (2) to yield that for any iteration $\widehat{x}^n = [\mathbf{F}^n + M_{t(n)}^{-1}] C^T R^{-1} y$, where the matrix $\mathbf{F}^n$ satisfies the recursion

$$\mathbf{F}^n = M_{t(n)}^{-1} K_{t(n)} \left[ \mathbf{F}^{n-1} + M_{t(n-1)}^{-1} \right] \tag{5}$$

with the initial condition $\mathbf{F}^1 = \mathbf{0}$. It is straightforward to show that whenever the recursion for the conditional means in equation (2) converges, then the matrix sequence $\{\mathbf{F}^n + M_{t(n)}^{-1}\}$ converges to the full error covariance $\widehat{P}$.

Moreover, the cutting matrices $K$ are typically of low rank, say $\mathcal{O}(E)$ where $E$ is the number of cut edges. On this basis, it can be shown that each $\mathbf{F}^n$ can be

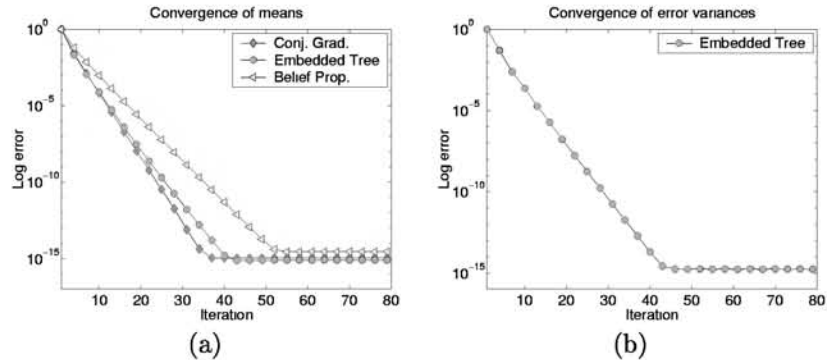

**Figure 2.** (a) Convergence rates for computing conditional means (normalized $L^2$ error). (b) Convergence rate of ET algorithm for computing error variances.

decomposed as a sum of $\mathcal{O}(E)$ rank 1 matrices. Directly updating this low–rank decomposition of $\mathbf{F}^n$ from that of $\mathbf{F}^{n-1}$ requires $\mathcal{O}(d^3 E^2 N)$ operations. However, an efficient restructuring of this update requires only $\mathcal{O}(d^3 E N)$ operations [11]. The diagonal blocks of the low–rank representation may be easily extracted and added to the diagonal blocks of $M_{t(n)}^{-1}$, which are computed by standard tree smoothers. All together, we may obtain these error variances in $\mathcal{O}(d^3 E N)$ operations per iteration. Thus, the computation of error variances will be particularly efficient for graphs where the number of edges $E$ that must be cut is small compared to the total number of nodes $N$.

### 3.4 Results

We have applied the algorithm to a variety of graphs, ranging from graphs with single loops to densely connected MRFs on grids. Figure 2(a) compares the rates of convergence for three algorithms: conjugate gradient (CG), embedded trees (ET), and belief propagation (BP) on a $20 \times 20$ nearest-neighbor grid. The ET algorithm employed two embedded trees analogous to those shown in Figure 1. We find that CG is usually fastest, and can exhibit supergeometric convergence. In accordance with Proposition 2, the ET algorithm converges geometrically. Either BP or ET can be made to converge faster, depending on the choice of clique potentials. However, we have not experimented with optimizing the performance of ET by adaptively choosing edges to cut. Figure 2(b) shows that in contrast to CG and BP, the ET algorithm can also be used to compute the conditional error variances, where the convergence rate is again geometric.

## 4 Modeling using graphs with cycles

### 4.1 Issues in model design

A variety of graphical structures may be used to approximate a given stochastic process. For example, perhaps the simplest model for a 1-D time series is a Markov chain, as shown in Figure 3(a). However, a high order model may be required to adequately capture long-range correlations. The associated increase in state dimension leads to inefficient estimation.

Figure 3(b) shows an alternative model structure. Here, additional "coarse scale" nodes have been added to the graph which are not directly linked to any measurements. These nodes are auxiliary variables created to explain the "fine scale" stochastic process of interest. If properly designed, the resulting tree structure

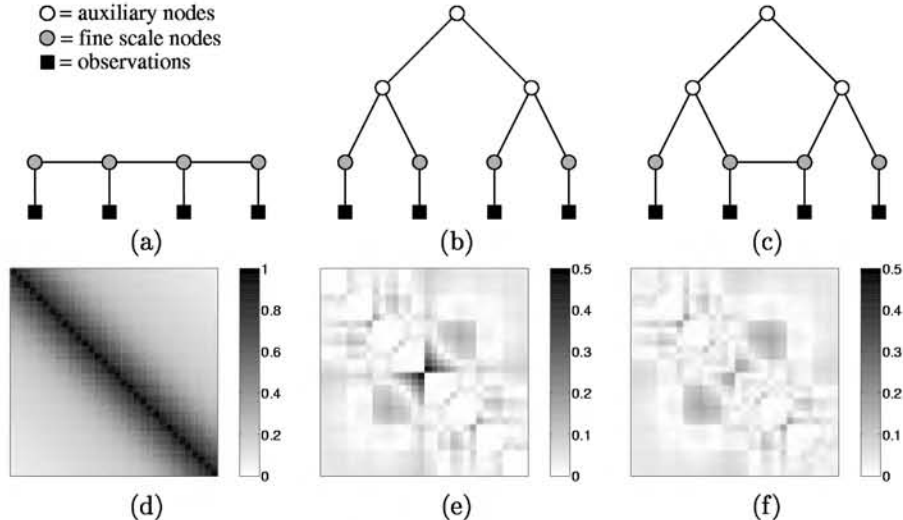

○ = auxiliary nodes
⬤ = fine scale nodes
■ = observations

(a)          (b)          (c)

(d)          (e)          (f)

**Figure 3.** (a) Markov chain. (b) Multiscale tree model. (c) Tree augmented by extra edge. (d) Desired covariance $P$. (e) Error $|P - P_{\text{tree}}|$ between desired covariance and realized tree covariance. (f) Error $|P - P_{\text{loop}}|$ between desired covariance and covariance realized with loopy graph.

will capture long-range correlations without the increase in state dimension of a higher-order Markov model. In previous work, our group has developed efficient algorithms for estimation and stochastic realization using such multiscale tree models [2, 4, 5, 12]. The gains provided by multiscale models are especially impressive when quadtrees are used to approximate two-dimensional Markov random fields. While statistical inference on MRFs is notoriously difficult, estimation on quadtrees remains extremely efficient.

The most significant weakness of tree models is boundary artifacts. That is, leaf nodes that are adjacent in the original process may be widely separated in the tree structure (see Figure 3(b)). As a result, dependencies between these nodes may be inadequately modeled, causing blocky discontinuities. Increasing the state dimension $d$ of the hidden nodes will reduce blockiness, but will also reduce estimation efficiency, which is $\mathcal{O}(d^3 N)$ in total. One potential solution is to add edges between pairs of fine scale nodes where tree artifacts are likely to arise, as shown in Figure 3(c). Such edges should be able to account for short-range dependency neglected by a tree model. Furthermore, optimal inference for such "near–tree" models using the ET algorithm will still be extremely efficient.

## 4.2 Application to multiscale modeling

Consider a one-dimensional process of length 32 with exact covariance $P$ shown in Figure 3(d). We approximate this process using two different graphical models, a multiscale tree and a "near-tree" containing an additional edge between two fine-scale nodes across a tree boundary (see Figure 3(c)). In both models, the state dimension at each node is constrained to be 2; therefore, the finest scale contains 16 nodes to model all 32 process points. Figure 3(e) shows the absolute error $|P - P_{\text{tree}}|$ for the tree model, where realization was performed by the scale-recursive algorithm presented in [12]. The tree model matches the desired process statistics relatively well except at the center, where the tree structure causes a boundary artifact. Figure 3(f) shows the absolute error $|P - P_{\text{loop}}|$ for a graph obtained by adding a single edge across the largest fine-scale tree boundary. The addition reduces the

peak error by 60%, a substantial gain in modeling fidelity. If the ET algorithm is implemented by cutting to two different embedded trees, it converges extremely rapidly with rate $\gamma = 0.11$.

## 5  Discussion

This paper makes contributions to both the estimation and modeling of Gaussian processes on graphs. First, we developed the embedded trees algorithm for estimation of Gaussian processes on arbitrary graphs. In contrast to other techniques, our algorithm computes both means and error covariances. Even on densely connected graphs, our algorithm is comparable to or better than other techniques for computing means. The error covariance computation is especially efficient for graphs in which cutting a small number of edges reveals an embedded tree. In this context, we have shown that modeling with sparsely connected loopy graphs can lead to substantial gains in modeling fidelity, with a minor increase in estimation complexity.

From the results of this paper arise a number of fundamental questions about the trade-off between modeling fidelity and estimation complexity. In order to address these questions, we are currently working to develop tighter bounds on the convergence rate of the algorithm, and also considering techniques for optimally selecting edges to be removed. On the modeling side, we are expanding on previous work for trees [12] in order to develop a theory of stochastic realization for processes on graphs with cycles. Lastly, although the current paper has focused on Gaussian processes, similar concepts can be developed for discrete-valued processes.

**Acknowledgments**

This work partially funded by ONR grant N00014-00-1-0089 and AFOSR grant F49620-98-1-0349; M.W. supported by NSERC 1967 fellowship, and E.S. by NDSEG fellowship.

## References

[1] J. Pearl. *Probabilistic reasoning in intelligent systems.* Morgan Kaufman, 1988.

[2] K. Chou, A. Willsky, and R. Nikoukhah. Multiscale systems, Kalman filters, and Riccati equations. *IEEE Trans. AC*, 39(3):479–492, March 1994.

[3] R. G. Gallager. *Low-density parity check codes.* MIT Press, Cambridge, MA, 1963.

[4] M. Luettgen, W. Karl, and A. Willsky. Efficient multiscale regularization with application to optical flow. *IEEE Trans. Im. Proc.*, 3(1):41–64, Jan. 1994.

[5] P. Fieguth, W. Karl, A. Willsky, and C. Wunsch. Multiresolution optimal interpolation of satellite altimetry. *IEEE Trans. Geo. Rem.*, 33(2):280–292, March 1995.

[6] P. Rusmevichientong and B. Van Roy. An analysis of turbo decoding with Gaussian densities. In *NIPS 12*, pages 575–581. MIT Press, 2000.

[7] Y. Weiss and W. T. Freeman. Correctness of belief propagation in Gaussian graphical models of arbitrary topology. In *NIPS 12*, pages 673–679. MIT Press, 2000.

[8] J. Besag. Spatial interaction and the statistical analysis of lattice systems. *J. Roy. Stat. Soc. Series B*, 36:192–236, 1974.

[9] J.W. Demmel. *Applied numerical linear algebra.* SIAM, Philadelphia, 1997.

[10] O. Axelsson. Bounds of eigenvalues of preconditioned matrices. *SIAM J. Matrix Anal. Appl.*, 13:847–862, July 1992.

[11] E. Sudderth, M. Wainwright, and A. Willsky. Embedded trees for modeling and estimation of Gaussian processes on graphs with cycles. In preparation, Dec. 2000.

[12] A. Frakt and A. Willsky. Computationally efficient stochastic realization for internal multiscale autoregressive models. *Mult. Sys. and Sig. Proc.* To appear.
